# Correcting Sample Selection Bias by Unlabeled Data

**Jiayuan Huang**
School of Computer Science
Univ. of Waterloo, Canada
*j9huang@cs.uwaterloo.ca*

**Alexander J. Smola**
NICTA, ANU
Canberra, Australia
*Alex.Smola@anu.edu.au*

**Arthur Gretton**
MPI for Biological Cybernetics
Tübingen, Germany
*arthur@tuebingen.mpg.de*

**Karsten M. Borgwardt**
Ludwig-Maximilians-University
Munich, Germany
*kb@dbs.ifi.lmu.de*

**Bernhard Schölkopf**
MPI for Biological Cybernetics
Tübingen, Germany
*bs@tuebingen.mpg.de*

## Abstract

We consider the scenario where training and test data are drawn from different distributions, commonly referred to as *sample selection bias*. Most algorithms for this setting try to first recover sampling distributions and then make appropriate corrections based on the distribution estimate. We present a nonparametric method which directly produces resampling weights without distribution estimation. Our method works by matching distributions between training and testing sets in feature space. Experimental results demonstrate that our method works well in practice.

## 1 Introduction

The default assumption in many learning scenarios is that training and test data are independently and identically (iid) drawn from the *same* distribution. When the distributions on training and test set do not match, we are facing *sample selection bias* or *covariate shift*. Specifically, given a domain of patterns $\mathcal{X}$ and labels $\mathcal{Y}$, we obtain training samples $Z = \{(x_1, y_1), \ldots, (x_m, y_m)\} \subseteq \mathcal{X} \times \mathcal{Y}$ from a Borel probability distribution $\Pr(x, y)$, and test samples $Z' = \{(x'_1, y'_1), \ldots, (x'_{m'}, y'_{m'})\} \subseteq \mathcal{X} \times \mathcal{Y}$ drawn from another such distribution $\Pr'(x, y)$.

Although there exists previous work addressing this problem [2, 5, 8, 9, 12, 16, 20], sample selection bias is typically ignored in standard estimation algorithms. Nonetheless, in reality the problem occurs rather frequently : While the available data have been collected in a biased manner, the test is usually performed over a more general target population. Below, we give two examples; but similar situations occur in many other domains.
1. Suppose we wish to generate a model to diagnose breast cancer. Suppose, moreover, that most women who participate in the breast screening test are middle-aged and likely to have attended the screening in the preceding 3 years. Consequently our sample includes mostly older women and those who have low risk of breast cancer because they have been tested before. The examples do not reflect the general population with respect to age (which amounts to a bias in $\Pr(x)$) and they only contain very few diseased cases (i.e. a bias in $\Pr(y|x)$).
2. Gene expression profile studies using DNA microarrays are used in tumor diagnosis. A common problem is that the samples are obtained using certain protocols, microarray platforms and analysis techniques. In addition, they typically have small sample sizes. The test cases are recorded under different conditions, resulting in a different distribution of gene expression values.

In this paper, we utilize the availability of unlabeled data to direct a sample selection de-biasing procedure for various learning methods. Unlike previous work we infer the resampling weight *directly* by distribution matching between training and testing sets in feature space in a non-parametric

manner. We do not require the estimation of biased densities or selection probabilities [20, 2, 12], or the assumption that probabilities of the different classes are known [8]. Rather, we account for the difference between $\Pr(x, y)$ and $\Pr'(x, y)$ by reweighting the training points such that the means of the training and test points in a reproducing kernel Hilbert space (RKHS) are close. We call this reweighting process kernel mean matching (KMM). When the RKHS is universal [14], the population solution to this miminisation is exactly the ratio $\Pr'(x, y)/\Pr(x, y)$; however, we also derive a cautionary result, which states that even granted this ideal population reweighting, the convergence of the empirical means in the RKHS depends on an upper bound on the ratio of distributions (but not on the dimension of the space), and will be extremely slow if this ratio is large.

The required optimisation is a simple QP problem, and the reweighted sample can be incorporated straightforwardly into several different regression and classification algorithms. We apply our method to a variety of regression and classification benchmarks from UCI and elsewhere, as well as to classification of microarrays from prostate and breast cancer patients. These experiments demonstrate that KMM greatly improves learning performance compared with training on unweighted data, and that our reweighting scheme can in some cases outperform reweighting using the true sample bias distribution.

**Key Assumption 1:** In general, the estimation problem with two different distributions $\Pr(x, y)$ and $\Pr'(x, y)$ is unsolvable, as the two terms could be arbitrarily far apart. In particular, for arbitrary $\Pr(y|x)$ and $\Pr'(y|x)$, there is no way we could infer a good estimator based on the training sample. Hence we make the simplifying assumption that $\Pr(x, y)$ and $\Pr'(x, y)$ only differ via $\Pr(x, y) = \Pr(y|x)\Pr(x)$ and $\Pr(y|x)\Pr'(x)$. In other words, the conditional probabilities of $y|x$ remain *unchanged* (this particular case of sample selection bias has been termed *covariate shift* [12]). However, we will see experimentally that even in situations where our key assumption is not valid, our method can nonetheless perform well (see Section 4).

## 2 Sample Reweighting

We begin by stating the problem of regularized risk minimization. In general a learning method minimizes the expected risk

$$R[\Pr, \theta, l(x, y, \theta)] = \mathbf{E}_{(x,y) \sim \Pr}[l(x, y, \theta)] \tag{1}$$

of a loss function $l(x, y, \theta)$ that depends on a parameter $\theta$. For instance, the loss function could be the negative log-likelihood $-\log \Pr(y|x, \theta)$, a misclassification loss, or some form of regression loss. However, since typically we only observe examples $(x, y)$ drawn from $\Pr(x, y)$ rather than $\Pr'(x, y)$, we resort to computing the empirical average

$$R_{\text{emp}}[Z, \theta, l(x, y, \theta)] = \frac{1}{m} \sum_{i=1}^{m} l(x_i, y_i, \theta). \tag{2}$$

To avoid overfitting, instead of minimizing $R_{\text{emp}}$ directly we often minimize a regularized variant $R_{\text{reg}}[Z, \theta, l(x, y, \theta)] := R_{\text{emp}}[Z, \theta, l(x, y, \theta)] + \lambda \Omega[\theta]$, where $\Omega[\theta]$ is a regularizer.

### 2.1 Sample Correction

The problem is more involved if $\Pr(x, y)$ and $\Pr'(x, y)$ are different. The training set is drawn from $\Pr$, however what we would really like is to minimize $R[\Pr', \theta, l]$ as we wish to generalize to test examples drawn from $\Pr'$. An observation from the field of importance sampling is that

$$R[\Pr', \theta, l(x, y, \theta)] = \mathbf{E}_{(x,y) \sim \Pr'}[l(x, y, \theta)] = \mathbf{E}_{(x,y) \sim \Pr}\left[\underbrace{\frac{\Pr'(x,y)}{\Pr(x,y)}}_{:=\beta(x,y)} l(x, y, \theta)\right] \tag{3}$$

$$= R[\Pr, \theta, \beta(x, y)l(x, y, \theta)], \tag{4}$$

*provided that the support of* $\Pr'$ *is contained in the support of* $\Pr$. Given $\beta(x, y)$, we can thus compute the risk with respect to $\Pr'$ using $\Pr$. Similarly, we can *estimate* the risk with respect to $\Pr'$ by computing $R_{\text{emp}}[Z, \theta, \beta(x, y)l(x, y, \theta)]$.

The key problem is that the coefficients $\beta(x, y)$ are usually unknown, and we need to estimate them from the data. When $\Pr$ and $\Pr'$ differ only in $\Pr(x)$ and $\Pr'(x)$, we have $\beta(x, y) = \Pr'(x)/\Pr(x)$, where $\beta$ is a reweighting factor for the training examples. We thus reweight every observation

$(x, y)$ such that observations that are under-represented in $\mathrm{Pr}$ obtain a higher weight, whereas over-represented cases are downweighted.

Now we could estimate $\mathrm{Pr}$ and $\mathrm{Pr}'$ and subsequently compute $\beta$ based on those estimates. This is closely related to the methods in [20, 8], as they have to either estimate the selection probabilities or have prior knowledge of the class distributions. Although intuitive, this approach has two major problems: first, it only works whenever the density estimates for $\mathrm{Pr}$ and $\mathrm{Pr}'$(or potentially, the selection probabilities or class distributions) are good. In particular, small errors in estimating $\mathrm{Pr}$ can lead to large coefficients $\beta$ and consequently to a serious overweighting of the corresponding observations. Second, estimating both densities just for the purpose of computing reweighting coefficients may be overkill: we may be able to directly estimate the coefficients $\beta_i := \beta(x_i, y_i)$ without having to estimate the two distributions. Furthermore, we can regularize $\beta_i$ directly with more flexibility, taking prior knowledge into account similar to learning methods for other problems.

## 2.2 Using the sample reweighting in learning algorithms

Before we describe how we will estimate the reweighting coefficients $\beta_i$, let us briefly discuss how to minimize the reweighted regularized risk

$$R_{\mathrm{reg}}[Z, \beta, l(x, y, \theta)] := \frac{1}{m} \sum_{i=1}^{m} \beta_i l(x_i, y_i, \theta) + \lambda \Omega[\theta], \tag{5}$$

in the classification and regression settings (an additional classification method is discussed in the accompanying technical report [7]).

**Support Vector Classification:** Utilizing the setting of [17]we can have the following minimization problem (the original SVMs can be formulated in the same way):

$$\underset{\theta, \xi}{\text{minimize}} \ \frac{1}{2} \|\theta\|^2 + C \sum_{i=1}^{m} \beta_i \xi_i \tag{6a}$$

$$\text{subject to } \langle \phi(x_i, y_i) - \phi(x_i, y), \theta \rangle \geq 1 - \xi_i / \Delta(y_i, y) \text{ for all } y \in \mathcal{Y}, \text{ and } \xi_i \geq 0. \tag{6b}$$

Here, $\phi(x, y)$ is a feature map from $\mathcal{X} \times \mathcal{Y}$ into a feature space $\mathcal{F}$, where $\theta \in \mathcal{F}$ and $\Delta(y, y')$ denotes a discrepancy function between $y$ and $y'$. The dual of (6) is given by

$$\underset{\alpha}{\text{minimize}} \ \frac{1}{2} \sum_{i,j=1; y, y' \in \mathcal{Y}}^{m} \alpha_{iy} \alpha_{jy'} k(x_i, y, x_j, y') - \sum_{i=1; y \in \mathcal{Y}}^{m} \alpha_{iy} \tag{7a}$$

$$\text{subject to } \alpha_{iy} \geq 0 \text{ for all } i, y \text{ and } \sum_{y \in \mathcal{Y}} \alpha_{iy} / \Delta(y_i, y) \leq \beta_i C. \tag{7b}$$

Here $k(x, y, x', y') := \langle \phi(x, y), \phi(x', y') \rangle$ denotes the inner product between the feature maps. This generalizes the observation-dependent binary SV classification described in [10]. Modifications of existing solvers, such as SVMStruct [17], are straightforward.

**Penalized LMS Regression:** Assume $l(x, y, \theta) = (y - \langle \phi(x), \theta \rangle)^2$ and $\Omega[\theta] = \|\theta\|^2$. Here we minimize

$$\sum_{i=1}^{m} \beta_i (y_i - \langle \phi(x_i), \theta \rangle)^2 + \lambda \|\theta\|^2. \tag{8}$$

Denote by $\bar{\beta}$ the diagonal matrix with diagonal $(\beta_1, \ldots, \beta_m)$ and let $K \in \mathbb{R}^{m \times m}$ be the kernel matrix $K_{ij} = k(x_i, x_j)$. In this case minimizing (8) is equivalent to minimizing $(y - K\alpha)^\top \bar{\beta}(y - K\alpha) + \lambda \alpha^\top K\alpha$ with respect to $\alpha$. Assuming that $K$ and $\bar{\beta}$ have full rank, the minimization yields $\alpha = (\lambda \bar{\beta}^{-1} + K)^{-1} y$. The advantage of this formulation is that it can be solved as easily as solving the standard penalized regression problem. Essentially, we rescale the regularizer depending on the pattern weights: the higher the weight of an observation, the less we regularize.

# 3 Distribution Matching

## 3.1 Kernel Mean Matching and its relation to importance sampling

Let $\Phi : \mathcal{X} \to \mathcal{F}$ be a map into a feature space $\mathcal{F}$ and denote by $\mu : \mathcal{P} \to \mathcal{F}$ the expectation operator

$$\mu(\mathrm{Pr}) := \mathbf{E}_{x \sim \mathrm{Pr}(x)} \left[ \Phi(x) \right]. \tag{9}$$

Clearly $\mu$ is a *linear* operator mapping the space of all probability distributions $\mathcal{P}$ into feature space. Denote by $\mathcal{M}(\Phi) := \{\mu(\mathrm{Pr})$ where $\mathrm{Pr} \in \mathcal{P}\}$ the image of $\mathcal{P}$ under $\mu$. This set is also often referred to as the *marginal polytope*. We have the following theorem (proved in [7]):

**Theorem 1** *The operator $\mu$ is* bijective *if $\mathcal{F}$ is an RKHS with a universal kernel $k(x, x') = \langle \Phi(x), \Phi(x') \rangle$ in the sense of Steinwart [15].*

The use of feature space means to compare distributions is further explored in [3]. The practical consequence of this (rather abstract) result is that if we know $\mu(\mathrm{Pr}')$, we can infer a suitable $\beta$ by solving the following minimization problem:

$$\underset{\beta}{\text{minimize}} \left\| \mu(\mathrm{Pr}') - \mathbf{E}_{x \sim \mathrm{Pr}(x)} \left[ \beta(x) \Phi(x) \right] \right\| \text{ subject to } \beta(x) \geq 0 \text{ and } \mathbf{E}_{x \sim \mathrm{Pr}(x)} \left[ \beta(x) \right] = 1. \tag{10}$$

This is the kernel mean matching (KMM) procedure. For a proof of the following (and further results in the paper) see [7].

**Lemma 2** *The problem (10) is convex. Moreover, assume that $\mathrm{Pr}'$ is absolutely continuous with respect to $\mathrm{Pr}$ (so $\mathrm{Pr}(A) = 0$ implies $\mathrm{Pr}'(A) = 0$). Finally assume that $k$ is universal. Then the solution $\beta(x)$ of (10) is $Pr'(x) = \beta(x) Pr(x)$.*

## 3.2 Convergence of reweighted means in feature space

Lemma 2 shows that in principle, if we knew $\mathrm{Pr}$ and $\mu[\mathrm{Pr}']$, we could fully recover $\mathrm{Pr}'$ by solving a simple quadratic program. In practice, however, neither $\mu(\mathrm{Pr}')$ nor $\mathrm{Pr}$ is known. Instead, we only have samples $X$ and $X'$ of size $m$ and $m'$, drawn iid from $\mathrm{Pr}$ and $\mathrm{Pr}'$ respectively.

Naively we could just replace the expectations in (10) by empirical averages and hope that the resulting optimization problem provides us with a good estimate of $\beta$. However, it is to be expected that empirical averages will differ from each other due to finite sample size effects. In this section, we explore two such effects. First, we demonstrate that in the finite sample case, for a fixed $\beta$, the empirical estimate of the expectation of $\beta$ is normally distributed: this provides a natural limit on the precision with which we should enforce the constraint $\int \beta(x) d\,\mathrm{Pr}(x) = 1$ when using empirical expectations (we will return to this point in the next section).

**Lemma 3** *If $\beta(x) \in [0, B]$ is some fixed function of $x \in \mathcal{X}$, then given $x_i \sim \mathrm{Pr}$ iid such that $\beta(x_i)$ has finite mean and non-zero variance, the sample mean $\frac{1}{m} \sum_i \beta(x_i)$ converges in distribution to a Gaussian with mean $\int \beta(x) d\,\mathrm{Pr}(x)$ and standard deviation bounded by $\frac{B}{2\sqrt{m}}$.*

This lemma is a direct consequence of the central limit theorem [1, Theorem 5.5.15]. Alternatively, it is straightforward to get a large deviation bound that likewise converges as $1/\sqrt{m}$ [6].

Our second result demonstrates the deviation between the empirical means of $\mathrm{Pr}'$ and $\beta(x) \mathrm{Pr}$ in feature space, given $\beta(x)$ is chosen perfectly in the population sense. In particular, this result shows that convergence of these two means will be slow if there is a large difference in the probability mass of $\mathrm{Pr}'$ and $\mathrm{Pr}$ (and thus the bound $B$ on the ratio of probability masses is large).

**Lemma 4** *In addition to the Lemma 3 conditions, assume that we draw $X' := \{x'_1, \ldots, x'_{m'}\}$ iid from $\mathcal{X}$ using $\mathrm{Pr}' = \beta(x) \mathrm{Pr}$, and $\|\Phi(x)\| \leq R$ for all $x \in \mathcal{X}$. Then with probability at least $1 - \delta$*

$$\left\| \frac{1}{m} \sum_{i=1}^m \beta(x_i) \Phi(x_i) - \frac{1}{m'} \sum_{i=1}^{m'} \Phi(x'_i) \right\| \leq \left( 1 + \sqrt{-2 \log \delta/2} \right) R \sqrt{B^2/m + 1/m'} \tag{11}$$

Note that this lemma shows that for a *given* $\beta(x)$, which is correct in the population sense, we can bound the deviation between the feature space mean of $\mathrm{Pr}'$ and the reweighted feature space mean of $\mathrm{Pr}$. It is *not* a guarantee that we will find coefficients $\beta_i$ that are close to $\beta(x_i)$, but it gives us a useful upper bound on the outcome of the optimization.

Lemma 4 implies that we have $O(B\sqrt{1/m + 1/m'B^2})$ convergence in $m, m'$ and $B$. This means that, for very different distributions we need a large equivalent sample size to get reasonable convergence. Our result also implies that it is unrealistic to assume that the empirical means (reweighted or not) should match exactly.

### 3.3  Empirical KMM optimization

To find suitable values of $\beta \in \mathbb{R}^m$ we want to minimize the discrepancy between means subject to constraints $\beta_i \in [0, B]$ and $|\frac{1}{m}\sum_{i=1}^{m}\beta_i - 1| \leq \epsilon$. The former limits the scope of discrepancy between $\mathrm{Pr}$ and $\mathrm{Pr}'$ whereas the latter ensures that the measure $\beta(x)\,\mathrm{Pr}(x)$ is close to a probability distribution. The objective function is given by the discrepancy term between the two empirical means. Using $K_{ij} := k(x_i, x_j)$ and $\kappa_i := \frac{m}{m'}\sum_{j=1}^{m'}k(x_i, x_j')$ one may check that

$$\left\| \frac{1}{m}\sum_{i=1}^{m}\beta_i\Phi(x_i) - \frac{1}{m'}\sum_{i=1}^{m'}\Phi(x_i')\right\|^2 = \frac{1}{m^2}\beta^\top K\beta - \frac{2}{m^2}\kappa^\top\beta + \text{const.}$$

We now have all necessary ingredients to formulate a quadratic problem to find suitable $\beta$ via

$$\underset{\beta}{\text{minimize}} \ \frac{1}{2}\beta^\top K\beta - \kappa^\top\beta \text{ subject to } \beta_i \in [0, B] \text{ and } \left|\sum_{i=1}^{m}\beta_i - m\right| \leq m\epsilon. \qquad (12)$$

In accordance with Lemma 3, we conclude that a good choice of $\epsilon$ should be $O(B/\sqrt{m})$. Note that (12) is a quadratic program which can be solved efficiently using interior point methods or any other successive optimization procedure. We also point out that (12) resembles Single Class SVM [11] using the $\nu$-trick. Besides the approximate equality constraint, the main difference is the linear correction term by means of $\kappa$. Large values of $\kappa_i$ correspond to particularly important observations $x_i$ and are likely to lead to large $\beta_i$.

## 4  Experiments

### 4.1  Toy regression example

Our first experiment is on toy data, and is intended mainly to provide a comparison with the approach of [12]. This method uses an information criterion to optimise the weights, under certain restrictions on $\mathrm{Pr}$ and $\mathrm{Pr}'$ (namely, $\mathrm{Pr}'$ must be known, while $\mathrm{Pr}$ can be either known exactly, Gaussian with unknown parameters, or approximated via kernel density estimation).

Our data is generated according to the polynomial regression example from [12, Section 2], for which $\mathrm{Pr} \sim \mathcal{N}(0.5, 0.5^2)$ and $\mathrm{Pr}' \sim \mathcal{N}(0, 0.3^2)$ are two normal distributions. The observations are generated according to $y = -x + x^3$, and are observed in Gaussian noise with standard deviation 0.3 (see Figure 1(a); the blue curve is the noise-free signal).

We sampled 100 training (blue circles) and testing (red circles) points from $\mathrm{Pr}$ and $\mathrm{Pr}'$ respectively. We attempted to model the observations with a degree 1 polynomial. The black dashed line is a best-case scenario, which is shown for reference purposes: it represents the model fit using ordinary least squared (OLS) on the labeled test points. The red line is a second reference result, derived only from the training data via OLS, and predicts the test data very poorly. The other three dashed lines are fit with weighted ordinary least square (WOLS), using one of three weighting schemes: the ratio of the underlying training and test densities, KMM, and the information criterion of [12]. A summary of the performance over 100 trials is shown in Figure 1(b). Our method outperforms the two other reweighting methods.

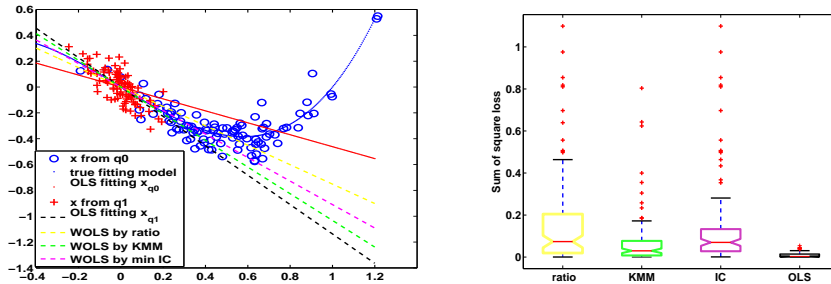

(a)                                  (b)

Figure 1: (a) Polynomial models of degree 1 fit with OLS and WOLS;(b) Average performances of three WOLS methods and OLS on the test data in (a). Labels are *Ratio* for ratio of test to training density; KMM for our approach; *min IC* for the approach of [12]; and *OLS* for the model trained on the labeled test points.

## 4.2 Real world datasets

We next test our approach on real world data sets, from which we select training examples using a deliberately biased procedure (as in [20, 9]). To describe our biased selection scheme, we need to define an additional random variable $s_i$ for each point in the pool of possible training samples, where $s_i = 1$ means the $i$th sample is included, and $s_i = 0$ indicates an excluded sample. Two situations are considered: the selection bias corresponds to our assumption regarding the relation between the training and test distributions, and $P(s_i = 1|x_i, y_i) = P(s_i|x_i)$; or $s_i$ is dependent only on $y_i$, i.e. $P(s_i|x_i, y_i) = P(s_i|y_i)$, which potentially creates a greater challenge since it violates our key assumption 1. In the following, we compare our method (labeled *KMM*) against two others: a baseline unweighted method (*unweighted*), in which no modification is made, and a weighting by the inverse of the true sampling distribution (*importance sampling*), as in [20, 9]. We emphasise, however, that our method does *not* require any prior knowledge of the true sampling probabilities. In our experiments, we used a Gaussian kernel $\exp(-\sigma\|x_i - x_j\|^2)$ in our kernel classification and regression algorithms, and parameters $\epsilon = (\sqrt{m} - 1)/\sqrt{m}$ and $B = 1000$ in the optimization (12).

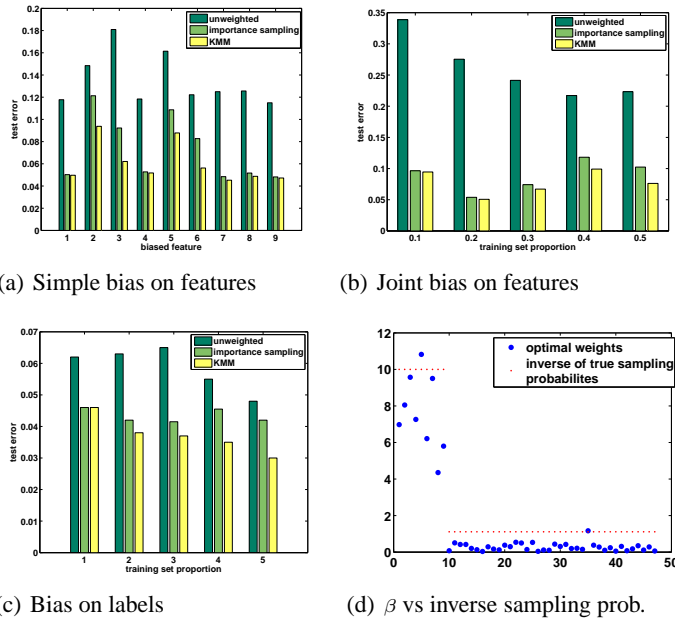

(a) Simple bias on features      (b) Joint bias on features

(c) Bias on labels      (d) $\beta$ vs inverse sampling prob.

Figure 2: Classification performance analysis on breast cancer dataset from UCI.

### 4.2.1 Breast Cancer Dataset

This dataset is from the UCI Archive, and is a binary classification task. It includes 699 examples from 2 classes: benign (positive label) and malignant (negative label). The data are randomly split into training and test sets, where the proportion of examples used for training varies from 10% to 50%. Test results are averaged over 30 trials, and were obtained using a support vector classifier with kernel size $\sigma = 0.1$. First, we consider a biased sampling scheme based on the input features, of which there are nine, with integer values from 0 to 9. Since smaller feature values predominate in the unbiased data, we sample according to $P(s = 1|x \le 5) = 0.2$ and $P(s = 1|x > 5) = 0.8$, repeating the experiment for each of the features in turn. Results are an average over 30 random training/test splits, with 1/4 of the data used for training and 3/4 for testing. Performance is shown in Figure 2(a): we consistently outperform the unweighted method, and match or exceed the performance obtained using the known distribution ratio. Next, we consider a sampling bias that operates jointly across multiple features. We select samples less often when they are further from the sample mean $\overline{x}$ over the training data, i.e. $P(s_i|x_i) \propto \exp(-\sigma\|x_i - \overline{x}\|^2)$ where $\sigma = 1/20$. Performance of our method in 2(b) is again better than the unweighted case, and as good as or better than reweighting using the sampling model. Finally, we consider a simple biased sampling scheme which depends only on the label $y$: $P(s = 1|y = 1) = 0.1$ and $P(s = 1|y = -1) = 0.9$ (the data has on average twice as many positive as negative examples when uniformly sampled). Average performance for different training/testing split proportions is in Figure 2(c); remarkably, despite our assumption regarding the difference between the training and test distributions being violated, our method still improves the test performance, and outperforms the reweighting by density ratio for large training set sizes. Fig-

ure 2(d) shows the weights $\beta$ are proportional to the inverse of true sampling probabilities: positive examples have higher weights and negative ones have lower weights.

### 4.2.2 Further Benchmark Datasets

We next compare the performance on further benchmark datasets[1] by selecting training data via various biased sampling schemes. Specifically, for the sampling distribution bias on labels, we use $P(s = 1|y) = \exp(a + by)/(1 + \exp(a + by))$ (datasets 1 to 5), or the simple step distribution $P(s = 1|y = 1) = a$, $P(s = 1|y = -1) = b$ (datasets 6 and 7). For the remaining datasets, we generate biased sampling schemes over their features. We first do PCA, selecting the first principal component of the training data and the corresponding projection values. Denoting the minimum value of the projection as $m$ and the mean as $\overline{m}$, we apply a normal distribution with mean $m + (\overline{m} - m)/a$ and variance $(\overline{m} - m)/b$ as the biased sampling scheme. Please refer to [7] for detailed parameter settings. We use penalized LMS for regression problems and SVM for classification problems. To evaluate generalization performance, we utilize the *normalized mean square error (NMSE)* given by $\frac{1}{n}\sum_{i=1}^{n}\frac{(y_i - \mu_i)}{\operatorname{var} y}$ for regression problems, and the average test error for classification problems. In 13 out of 23 experiments, our reweighting approach is the most accurate (see Table 1), despite having no prior information about the bias of the test sample (and, in some cases, despite the additional fact that the data reweighting does not conform to our key assumption 1). In addition, the KMM *always* improves test performance compared with the unweighted case. Two additional points should be borne in mind: first, we use the same $\sigma$ for the kernel mean matching and the SVM, as listed in Table 1. Performance might be improved by decoupling these kernel sizes: indeed, we employ kernels that are somewhat large, suggesting that the KMM procedure is helpful in the case of relatively smooth classification/regresssion functions. Second, we did not find a performance improvement in the case of data sets with smaller sample sizes. This is not surprising, since a reweighting would further reduce the effective number of points used for training, resulting in insufficient data for learning.

Table 1: Test results for three methods on 18 datasets with different sampling schemes. The results are averages over 10 trials for regression problems (marked *) and 30 trials for classification problems. We used a Gaussian kernel of size $\sigma$ for both the kernel mean matching and the SVM/LMS regression, and set $B = 1000$.

| DataSet | $\sigma$ | $n_{tr}$ | selected | $n_{tst}$ | unweighted | NMSE / Test err. importance samp. | KMM |
|---|---|---|---|---|---|---|---|
| 1. Abalone* | $1e-1$ | 2000 | 853 | 2177 | $1.00 \pm 0.08$ | $1.1 \pm 0.2$ | $\mathbf{0.6 \pm 0.1}$ |
| 2. CA Housing* | $1e-1$ | 16512 | 3470 | 4128 | $2.29 \pm 0.01$ | $1.72 \pm 0.04$ | $\mathbf{1.24 \pm 0.09}$ |
| 3. Delta Ailerons(1)* | $1e3$ | 4000 | 1678 | 3129 | $0.51 \pm 0.01$ | $0.51 \pm 0.01$ | $\mathbf{0.401 \pm 0.007}$ |
| 4. Ailerons* | $1e-5$ | 7154 | 925 | 6596 | $1.50 \pm 0.06$ | $\mathbf{0.7 \pm 0.1}$ | $1.2 \pm 0.2$ |
| 5. haberman(1) | $1e-2$ | 150 | 52 | 156 | $0.50 \pm 0.09$ | $0.37 \pm 0.03$ | $\mathbf{0.30 \pm 0.05}$ |
| 6. USPS(6vs8)(1) | $1/128$ | 500 | 260 | 1042 | $0.13 \pm 0.18$ | $\mathbf{0.1 \pm 0.2}$ | $0.1 \pm 0.1$ |
| 7. USPS(3vs9)(1) | $1/128$ | 500 | 252 | 1145 | $0.016 \pm 0.006$ | $\mathbf{0.012 \pm 0.005}$ | $0.013 \pm 0.005$ |
| 8. Bank8FM* | $1e-1$ | 4500 | 654 | 3692 | $0.5 \pm 0.1$ | $\mathbf{0.45 \pm 0.06}$ | $0.47 \pm 0.05$ |
| 9. Bank32nh* | $1e-2$ | 4500 | 740 | 3692 | $23 \pm 4.0$ | $\mathbf{19 \pm 2}$ | $\mathbf{19 \pm 2}$ |
| 10. cpu-act* | $1e-12$ | 4000 | 1462 | 4192 | $10 \pm 1$ | $4.0 \pm 0.2$ | $\mathbf{1.9 \pm 0.2}$ |
| 11. cpu-small* | $1e-12$ | 4000 | 1488 | 4192 | $9 \pm 2$ | $4.0 \pm 0.2$ | $\mathbf{2.0 \pm 0.5}$ |
| 12. Delta Ailerons(2)* | $1e3$ | 4000 | 634 | 3129 | $2 \pm 2$ | $\mathbf{1.5 \pm 1.5}$ | $1.7 \pm 0.9$ |
| 13. Boston house* | $1e-4$ | 300 | 108 | 206 | $0.8 \pm 0.2$ | $\mathbf{0.74 \pm 0.09}$ | $0.76 \pm 0.07$ |
| 14. kin8nm* | $1e-1$ | 5000 | 428 | 3192 | $0.85 \pm 0.2$ | $\mathbf{0.81 \pm 0.1}$ | $0.81 \pm 0.2$ |
| 15. puma8nh* | $1e-1$ | 4499 | 823 | 3693 | $1.1 \pm 0.1$ | $\mathbf{0.77 \pm 0.05}$ | $0.83 \pm 0.03$ |
| 16. haberman(2) | $1e-2$ | 150 | 90 | 156 | $0.27 \pm 0.01$ | $0.39 \pm 0.04$ | $\mathbf{0.25 \pm 0.2}$ |
| 17. USPS(6vs8) (2) | $1/128$ | 500 | 156 | 1042 | $0.23 \pm 0.2$ | $0.23 \pm 0.2$ | $\mathbf{0.16 \pm 0.08}$ |
| 18. USPS(6vs8) (3) | $1/128$ | 500 | 104 | 1042 | $0.54 \pm 0.0002$ | $0.5 \pm 0.2$ | $\mathbf{0.16 \pm 0.04}$ |
| 19. USPS(3vs9)(2) | $1/128$ | 500 | 252 | 1145 | $0.46 \pm 0.09$ | $0.5 \pm 0.2$ | $\mathbf{0.2 \pm 0.1}$ |
| 20. Breast Cancer | $1e-1$ | 280 | 96 | 419 | $0.05 \pm 0.01$ | $0.036 \pm 0.005$ | $\mathbf{0.033 \pm 0.004}$ |
| 21. India diabetes | $1e-4$ | 200 | 97 | 568 | $0.32 \pm 0.02$ | $\mathbf{0.30 \pm 0.02}$ | $\mathbf{0.30 \pm 0.02}$ |
| 22. ionosphere | $1e-1$ | 150 | 64 | 201 | $0.32 \pm 0.06$ | $0.31 \pm 0.07$ | $\mathbf{0.28 \pm 0.06}$ |
| 23. German credit | $1e-4$ | 400 | 214 | 600 | $0.283 \pm 0.004$ | $0.282 \pm 0.004$ | $\mathbf{0.280 \pm 0.004}$ |

### 4.2.3 Tumor Diagnosis using Microarrays

Our next benchmark is a dataset of 102 microarrays from prostate cancer patients [13]. Each of these microarrays measures the expression levels of 12,600 genes. The dataset comprises 50 samples from normal tissues (positive label) and 52 from tumor tissues (negative label). We simulate the realisitc scenario that two sets of microarrays A and B are given with dissimilar proportions of tumor samples, and we want to perform cancer diagnosis via classification, training on A and predicting

on B. We select training examples via the biased selection scheme $P(s = 1|y = 1) = 0.85$ and $P(s = 1|y = -1) = 0.15$. The remaining data points form the test set. We then perform SVM classification for the unweighted, KMM, and importance sampling approaches. The experiment was repeated over 500 independent draws from the dataset according to our biased scheme; the 500 resulting test errors are plotted in [7]. The KMM achieves much higher accuracy levels than the unweighted approach, and is very close to the importance sampling approach.

We study a very similar scenario on two breast cancer microarray datasets from [4] and [19], measuring the expression levels of 2,166 common genes for normal and cancer patients [18]. We train an SVM on one of them and test on the other. Our reweighting method achieves significant improvement in classification accuracy over the unweighted SVM (see [7]). Hence our method promises to be a valuable tool for cross-platform microarray classification.

**Acknowledgements:** The authors thank Patrick Warnat (DKFZ, Heidelberg) for providing the microarray datasets, and Olivier Chapelle and Matthias Hein for helpful discussions. The work is partially supported by by the BMBF under grant 031U112F within the BFAM project, which is part of the German Genome Analysis Network. NICTA is funded through the Australian Government's *Backing Australia's Ability* initiative, in part through the ARC. This work was supported in part by the IST Programme of the EC, under the PASCAL Network of Excellence, IST-2002-506778.

## Footnotes

[1]Regression data from `http://www.liacc.up.pt/~ltorgo/Regression/DataSets.html`; classification data from UCI. Sets with numbers in brackets are examined by different sampling schemes.

# References

[1] G. Casella and R. Berger. *Statistical Inference*. Duxbury, Pacific Grove, CA, 2nd edition, 2002.

[2] M. Dudik, R.E. Schapire, and S.J. Phillips. Correcting sample selection bias in maximum entropy density estimation. In *Advances in Neural Information Processing Systems 17*, 2005.

[3] A. Gretton, K. Borgwardt, M. Rasch, B. Schölkopf, and A. Smola. A kernel method for the two-sample-problem. In *NIPS*. MIT Press, 2006.

[4] S. Gruvberger, M. Ringner, Y.Chen, S.Panavally, L.H. Saal, C. Peterson A.Borg, M. Ferno, and P.S.Meltzer. Estrogen receptor status in breast cancer is associated with remarkably distinct gene expression patterns. *Cancer Research*, 61, 2001.

[5] J. Heckman. Sample selection bias as a specification error. *Econometrica*, 47(1):153–161, 1979.

[6] W. Hoeffding. Probability inequalities for sums of bounded random variables. *Journal of the American Statistical Association*, 58:13–30, 1963.

[7] J. Huang, A. Smola, A. Gretton, K. Borgwardt, and B. Schölkopf. Correcting sample selection bias by unlabeled data. Technical report, CS-2006-44, University of Waterloo, 2006.

[8] Y. Lin, Y. Lee, and G. Wahba. Support vector machines for classification in nonstandard situations. *Machine Learning*, 46:191–202, 2002.

[9] S. Rosset, J. Zhu, H. Zou, and T. Hastie. A method for inferring label sampling mechanisms in semi-supervised learning. In *Advances in Neural Information Processing Systems 17*, 2004.

[10] M. Schmidt and H. Gish. Speaker identification via support vector classifiers. In *Proc. ICASSP '96*, pages 105–108, Atlanta, GA, May 1996.

[11] B. Schölkopf, J. Platt, J. Shawe-Taylor, A. J. Smola, and R. C. Williamson. Estimating the support of a high-dimensional distribution. *Neural Computation*, 13(7):1443–1471, 2001.

[12] H. Shimodaira. Improving predictive inference under convariance shift by weighting the log-likelihood function. *Journal of Statistical Planning and Inference*, 90, 2000.

[13] D. Singh, P. Febbo, K. Ross, D. Jackson, J. Manola, C. Ladd, P. Tamayo, A. Renshaw, A. DAmico, and J. Richie. Gene expression correlates of clinical prostate cancer behavior. *Cancer Cell*, 1(2), 2002.

[14] I. Steinwart. On the influence of the kernel on the consistency of support vector machines. *Journal of Machine Learning Research*, 2:67–93, 2002.

[15] I. Steinwart. Support vector machines are universally consistent. *J. Compl.*, 18:768–791, 2002.

[16] M. Sugiyama and K.-R. Müller. Input-dependent estimation of generalization error under covariate shift. *Statistics and Decisions*, 23:249–279, 2005.

[17] I. Tsochantaridis, T. Joachims, T. Hofmann, and Y. Altun. Large margin methods for structured and interdependent output variables. *Journal of Machine Learning Research*, 2005.

[18] P. Warnat, R. Eils, and B. Brors. Cross-platform analysis of cancer microarray data improves gene expression based classification of phenotypes. *BMC Bioinformatics*, 6:265, Nov 2005.

[19] M. West, C. Blanchette, H. Dressman, E. Huang, S. Ishida, R. Spang, H Zuzan, J.A. Olson Jr, J.R.Marks, and J.R.Nevins. Predicting the clinical status of human breast cancer by using gene expression profiles. *PNAS*, 98(20), 2001.

[20] B. Zadrozny. Learning and evaluating classifiers under sample selection bias. In *International Conference on Machine Learning ICML'04*, 2004.
